# Computation of Heading Direction From Optic Flow in Visual Cortex

**Markus Lappe\***       **Josef P. Rauschecker**

Laboratory of Neurophysiology, NIMH, Poolesville, MD, U.S.A. and
Max–Planck–Institut für Biologische Kybernetik, Tübingen, Germany

## Abstract

We have designed a neural network which detects the direction of ego–motion from optic flow in the presence of eye movements (Lappe and Rauschecker, 1993). The performance of the network is consistent with human psychophysical data, and its output neurons show great similarity to "triple component" cells in area MSTd of monkey visual cortex. We now show that by using assumptions about the kind of eye movements that the observer is likely to perform, our model can generate various other cell types found in MSTd as well.

## 1   INTRODUCTION

Following the ideas of Gibson in the 1950's a number of studies in human psychophysics have demonstrated that optic flow can be used effectively for navigation in space (Rieger and Toet, 1985; Stone and Perrone, 1991; Warren *et al.*, 1988). In search for the neural basis of optic flow processing, an area in the cat's extrastriate visual cortex (PMLS) was described as having a centrifugal organization of neuronal direction preferences, which suggested an involvement of area PMLS in the processing of expanding flow fields (Rauschecker *et al.*, 1987; Brenner and Rauschecker, 1990). Recently, neurons in the dorsal part of the medial superior temporal area (MSTd) in monkeys have been described that respond to various combinations of large expanding/contracting, rotating, or shifting dot patterns (Duffy and Wurtz, 1991; Tanaka and Saito, 1989). Cells in MSTd show a continuum of response properties ranging from selectivity for only one movement pattern ("single

component cells") to selectivity for one mode of each of the three movement types ("triple component cells"). An interesting property of many MSTd cells is their position invariance (Andersen *et al.*, 1990). A sizable proportion of cells, however, do change their selectivity when the stimulus is displaced by several tens of degrees of visual angle, and their position dependence seems to be correlated with the type of movement selectivity (Duffy and Wurtz, 1991; Orban *et al.*, 1992): It is most common for triple component cells and occurs least often in single component cells. Taken together, the wide range of directional tuning and the apparent lack of specificity for the spatial position of a stimulus seem to suggest, that MSTd cells do not possess the selectivity needed to explain the high accuracy of human observers in psychophysical experiments. Our simulation results, however, demonstrate that a population encoding can be used, in which individual neurons are rather broadly tuned while the whole network gives very accurate results.

## 2    THE NETWORK MODEL

The major projections to area MST originate from the middle temporal area (MT). Area MT is a well known area of monkey cortex specialized for the processing of visual motion. It contains a retinotopic representation of local movement directions (Allman and Kaas, 1971; Maunsell and Van Essen, 1983). In our model we assume that area MT comprises a population encoding of the optic flow and that area MST uses this input from MT to extract the heading direction. Therefore, the network consists of two layers. In the first layer, 300 optic flow vectors at random locations within 50 degrees of eccentricity are represented. Each flow vector is encoded by a population of directionally selective neurons. It has been shown previously that a biologically plausible population encoding like this can also be modelled by a neural network (Wang *et al.*, 1989). For simplicity we use only four neurons to represent an optic flow vector $\theta_i$ as

$$\theta_i = \sum_{k=1}^{4} s_{ik} e_{ik}, \tag{1}$$

with equally spaced preferred directions $e_{ik} = (\cos(\pi k/2), \sin(\pi k/2))^t$. A neuron's response to a flow vector of direction $\phi_i$ and speed $\theta_i$ is given by the tuning curve

$$s_{ik} = \begin{cases} \theta_i \cos(\phi_i - \pi k/2) & \text{if } \cos(\phi_i - \pi k/2) > 0 \\ 0 & \text{otherwise.} \end{cases}$$

The second layer contains a retinotopic grid of possible translational heading directions $\mathbf{T}_j$. Each direction is represented by a population of neurons, whose summed activities give the likelihood that $\mathbf{T}_j$ is the correct heading. The perceived direction is finally chosen to be the one that has the highest population activity.

The calculation of this likelihood is based on the subspace algorithm by Heeger and Jepson (1992). It employs the minimization of a residual function over all possible heading directions. The neuronal populations in the second layer evaluate a related function that is maximal for the correct heading. The subspace algorithm works as follows: When an observer moves through a static environment all points in space share the same six motion parameters, the translation $\mathbf{T} = (T_x, T_y, T_z)^t$ and the rotation $\Omega = (\Omega_x, \Omega_y, \Omega_z)^t$. The optic flow $\theta(x, y)$ is the projection of the movement of a 3D–point $(X, Y, Z)^t$ onto the retina, which, for simplicity, is modelled as an image plane. In a viewer centered coordinate

system the optic flow can be written as:

$$\theta(x, y) = \frac{1}{Z(x, y)} \mathbf{A}(x, y)\mathbf{T} + \mathbf{B}(x, y)\mathbf{\Omega} \tag{2}$$

with the matrices

$$\mathbf{A}(x, y) = \begin{pmatrix} -f & 0 & x \\ 0 & -f & y \end{pmatrix} \quad \text{and} \quad \mathbf{B}(x, y) = \begin{pmatrix} xy/f & -f - x^2/f & y \\ f + y^2/f & -xy/f & -x \end{pmatrix}$$

depending only on coordinates $(x, y)$ in the image plane and on the "focal length" $f$ (Heeger and Jepson, 1992). In trying to estimate $\mathbf{T}$, given the optic flow $\theta$, we first have to note that the unknowns $Z(x, y)$ and $\mathbf{T}$ are multiplied together. They can thus not be determined independently so that the translation is considered a unit vector pointing in the direction of heading. Eq. (2) now contains six unknowns, $Z(x, y)$, $\mathbf{T}$ and $\mathbf{\Omega}$, but only two measurements $\theta_x$ and $\theta_y$. Therefore, flow vectors from $m$ distinct image points are combined into the matrix equation

$$\mathbf{\Theta} = \mathbf{C}(\mathbf{T})\mathbf{q}, \tag{3}$$

where $\mathbf{\Theta} = (\theta_1, \ldots, \theta_m)^t$ is a $2m$–dimensional vector consisting of the components of the $m$ image velocities, $\mathbf{q} = (1/Z(x_1, y_1), \ldots, 1/Z(x_m, y_m), \Omega_x, \Omega_y, \Omega_z)^t$ an $(m + 3)$–dimensional vector, and

$$\mathbf{C}(\mathbf{T}) = \begin{pmatrix} \mathbf{A}(x_1, y_1)\mathbf{T} & \cdots & 0 & \mathbf{B}(x_1, y_1) \\ \vdots & \ddots & \vdots & \vdots \\ 0 & \cdots & \mathbf{A}(x_m, y_m)\mathbf{T} & \mathbf{B}(x_m, y_m) \end{pmatrix} \tag{4}$$

a $2m \times (m + 3)$ matrix. Heeger and Jepson (1992) show that the heading direction can be recovered by minimizing the residual function

$$\mathrm{R}(\mathbf{T}) = \|\mathbf{\Theta}^t \mathbf{C}^\perp(\mathbf{T})\|^2.$$

In this equation $\mathbf{C}^\perp(\mathbf{T})$ is defined as follows: Provided that the columns of $\mathbf{C}(\mathbf{T})$ are linearly independent, they form a basis of an $(m + 3)$–dimensional subspace of the $\mathcal{R}^{2m}$, which is called the range of $\mathbf{C}(\mathbf{T})$. The matrix $\mathbf{C}^\perp(\mathbf{T})$ spans the remaining $(2m - (m+3))$–dimensional subspace which is called the orthogonal complement of $\mathbf{C}(\mathbf{T})$. Every vector in the orthogonal complement of $\mathbf{C}(\mathbf{T})$ is orthogonal to every vector in the range of $\mathbf{C}(\mathbf{T})$.

In the network, the population of neurons representing a certain $\mathbf{T}_j$ shall be maximally excited when $\mathrm{R}(\mathbf{T}_j) = 0$. Two steps are necessary to accomplish this. First an individual neuron evaluates part of the argument of $\mathrm{R}(\mathbf{T}_j)$ by picking out one of the column vectors of $\mathbf{C}^\perp(\mathbf{T}_j)$, denoted by $\mathbf{C}_l^\perp(\mathbf{T}_j)$, and computing $\mathbf{\Theta}^t \mathbf{C}_l^\perp(\mathbf{T}_j)$. This is done in the following way: $m$ first layer populations are chosen to form the neuron's input receptive field. The neuron's output is given by the sigmoid function

$$u_{jl} = g(\sum_{i=1}^{m} \sum_{k=1}^{4} J_{ijkl} s_{ik} - \mu), \tag{5}$$

in which $J_{ijkl}$ denotes the strength of the synaptic connection between the $l$–th output neuron in the second layer population representing heading direction $\mathbf{T}_j$ and the $k$–th input neuron in the first layer population representing the optic flow vector $\theta_i$, $\mu$ denotes the threshold. For the synaptic strengths we require that:

$$\sum_{i=1}^{m} \sum_{k=1}^{4} J_{ijkl} s_{ik} = \mathbf{\Theta}^t \mathbf{C}_l^\perp(\mathbf{T}_j). \tag{6}$$

At a single image location $i$ this is:

$$\sum_{k=1}^{4} J_{ijkl}s_{ik} = \theta_i^t \begin{pmatrix} C_{l,2i-1}^{\perp}(\mathbf{T}_j) \\ C_{l,2i}^{\perp}(\mathbf{T}_j) \end{pmatrix}.$$

Substituting eq. (1) we find:

$$\sum_{k=1}^{4} J_{ijkl}s_{ik} = \sum_{k=1}^{4} s_{ik}e_{ik}^t \begin{pmatrix} C_{l,2i-1}^{\perp}(\mathbf{T}_j) \\ C_{l,2i}^{\perp}(\mathbf{T}_j) \end{pmatrix}.$$

Therefore we set the synaptic strengths to:

$$J_{ijkl} = e_{ik}^t \begin{pmatrix} C_{l,2i-1}^{\perp}(\mathbf{T}_j) \\ C_{l,2i}^{\perp}(\mathbf{T}_j) \end{pmatrix}.$$

Then, whenever $\mathbf{T}_j$ is compatible with the measured optic flow, i.e. when $\Theta$ is in the range of $C(\mathbf{T}_j)$, the neuron receives a net input of zero. In the second step, another neuron $u_{jl'}$ is constructed so that the sum of the activities of the two neurons is maximal in this situation. Both neurons are connected to the same set of image locations but their connection strengths satisfy $J_{ijkl'} = -J_{ijkl}$. In addition, the threshold $\mu$ is given a slightly negative value. Then both their sigmoid transfer functions overlap at zero input, and the sum has a single peak. Finally, the neurons in every second layer population are organized in such matched pairs so that each population $j$ generates its maximal activity when $R(\mathbf{T}_j) = 0$.

In simulations, our network is able to compute the direction of heading with a mean error of less than one degree in agreement with human psychophysical data (see Lappe and Rauschecker, 1993). Like heading detection in human observers it functions over a wide range of speeds, it works with sparse flow fields and it needs depth in the visual environment when eye movements are performed.

## 3   DIFFERENT RESPONSE SELECTIVITIES

For the remainder of this paper we will focus on the second layer neuron's response properties by carrying out simulations analogous to neurophysiological experiments (Andersen *et al.*, 1990; Duffy and Wurtz, 1991; Orban *et al.*, 1992). A single neuron is constructed that receives input from 30 random image locations forming a $60 \times 60$ degree receptive field. The receptive field occupies the lower left quadrant of the visual field and also includes the fovea (Fig. 1A). The neuron is then presented with shifting, expanding/contracting and rotating optic flow patterns. The center $(x_c, y_c)$ of the expanding/contracting and rotating patterns is varied over the $100 \times 100$ degree visual field in order to test the position dependence of the neuron's responses. Directional tuning is assessed via the direction $\Phi$ of the shifting patterns. All patterns are obtained by choosing suitable translations and rotations in eq. (2). For instance, rotating patterns centered at $(x_c, y_c)$ are generated by

$$\mathbf{T} = 0 \quad \text{and} \quad \Omega = \frac{\pm\Omega}{\sqrt{x_c^2 + y_c^2 + f^2}} \begin{pmatrix} x_c \\ y_c \\ f \end{pmatrix}. \tag{7}$$

In keeping with the most common experimental condition, all depth values $Z(x_i, y_i)$ are taken to be equal.

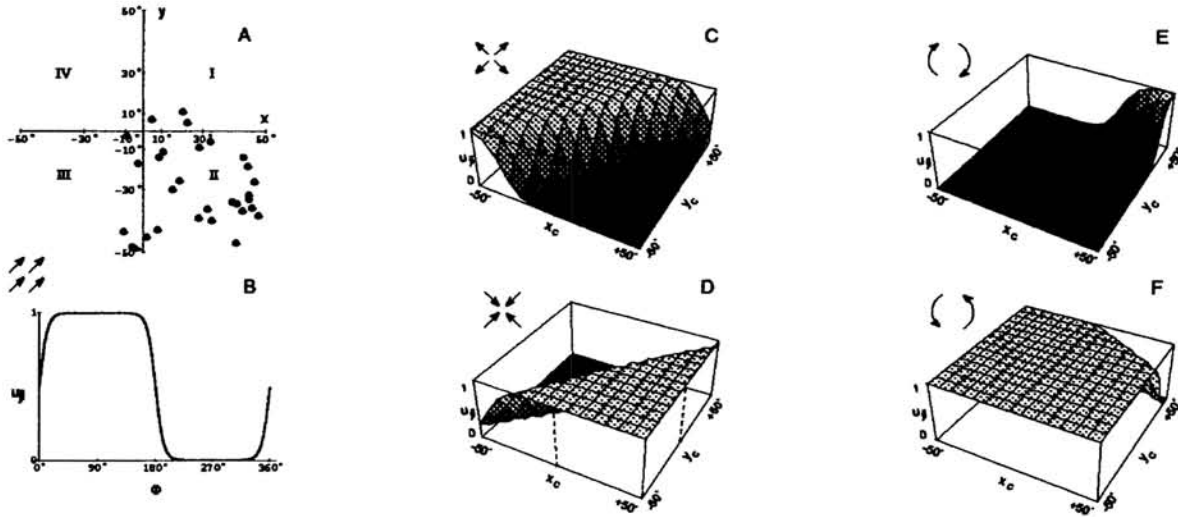

Figure 1: Single Neuron Responding To All Three Types Of Optic Flow Stimuli ("Triple Component Cell")

In the following we consider different assumptions about the observer's eye movements. These assumptions change the equations of the subspace algorithm. The rotational matrix $\mathbf{B}(x, y)$ takes on different forms. We will show that these changes result in different cell types. First let us restrict the model to the biologically most important case: During locomotion in a static environment the eye movements of humans or higher animals are usually the product of intentional behavior. A very common situation is the fixation of a visible object during locomotion. A specific eye rotation is necessary to compensate for the translational body–movement and to keep the object fixed in the center $(0, 0)$ of the visual field, so that its image velocity eq. (2) vanishes:

$$\boldsymbol{\theta}(0,0) = \frac{1}{Z_F} \begin{pmatrix} -fT_X \\ -fT_Y \end{pmatrix} + \begin{pmatrix} -f\Omega_Y \\ +f\Omega_X \end{pmatrix} = \begin{pmatrix} 0 \\ 0 \end{pmatrix}. \tag{8}$$

$Z_F$ denotes the distance of the fixation point. We can easily calculate $\Omega_X$ and $\Omega_Y$ from eq. (8) and chose $\Omega_Z = 0$. The optic flow eq. (2) in the case of the fixation of a stationary object then is

$$\tilde{\boldsymbol{\theta}}(x, y) = \frac{1}{Z(x, y)} \mathbf{A}(x, y)\mathbf{T} + \frac{1}{Z_F} \tilde{\mathbf{B}}(x, y)\mathbf{T},$$

with

$$\tilde{\mathbf{B}}(x, y) = \begin{pmatrix} f + x^2/f & (xy)/f & 0 \\ (xy)/f & f + y^2/f & 0 \end{pmatrix}.$$

We would like to emphasize that another common situation, namely no eye movements at all, can be approximated by $Z_F \to \infty$. We can now construct a new matrix

$$\tilde{\mathbf{C}}(\mathbf{T}) = \begin{pmatrix} \mathbf{A}(x_1, y_1)\mathbf{T} & \cdots & 0 & \tilde{\mathbf{B}}(x_1, y_1)\mathbf{T} \\ \vdots & \ddots & \vdots & \vdots \\ 0 & \cdots & \mathbf{A}(x_m, y_m)\mathbf{T} & \tilde{\mathbf{B}}(x_m, y_m)\mathbf{T} \end{pmatrix}$$

and form synaptic connections in the same way as described above. The resulting network is able to deal with the most common types of eye movements. The response properties of a

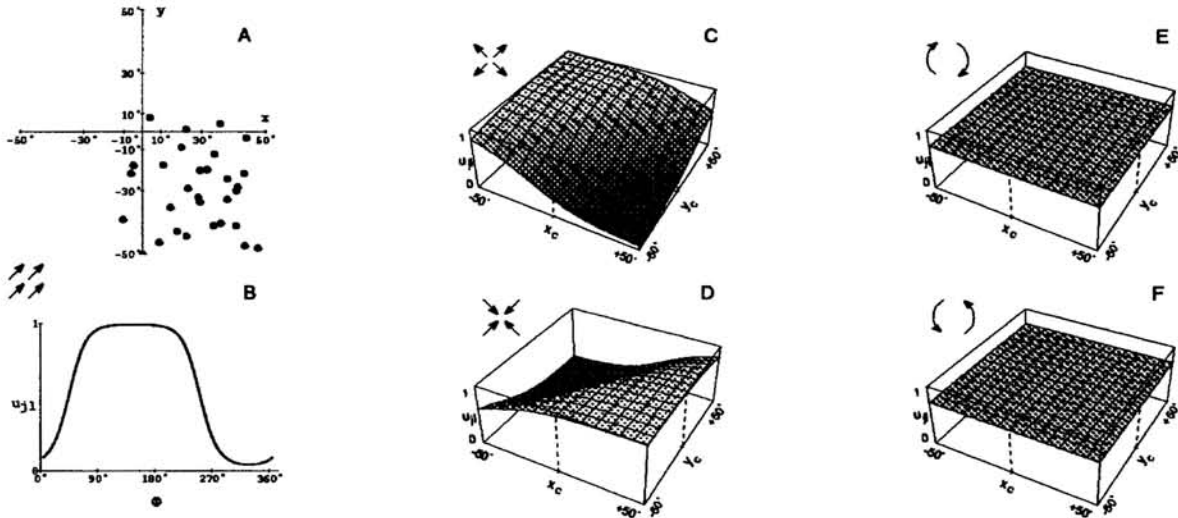

Figure 2: Neuron Selective For Two Components ("Double Component Cell")

single neuron from such a network are shown in Fig. 1. The neuron is selective for all three types of flow patterns. It exhibits broad directional tuning (Fig. 1B) for upward shifting patterns ($\Phi = 90$ deg.). The responses to expanding (Fig. 1C), contracting (Fig. 1D) and rotating (Fig. 1E–F) patterns show large areas of position invariant selectivity. Inside the receptive field, which covers the second quadrant (see distribution of input locations in Fig. 1A), the neuron favors upward shifts, contractions and counterclockwise rotations. It is thus compatible with a triple component cell in MSTd. Also, lines are visible along which the selectivities reverse. This happens because the neuron's input is a linear function of the stimulus position $(x_c, y_c)$. For example, for rotational patterns we can calculate the input using eqs. (2), (6), and (7):

$$\sum_{i=1}^{m}\sum_{k=1}^{4} J_{ijkl}s_{ik} = \frac{\pm\Omega}{\sqrt{x_c^2 + y_c^2 + f^2}} \sum_{i=1}^{m} \frac{1}{Z_F}(x_c, y_c, f)\tilde{\mathbf{B}}^t(x_i, y_i)\begin{pmatrix} C_{l,2i-1}^{\perp}(\mathbf{T}_j) \\ C_{l,2i}^{\perp}(\mathbf{T}_j) \end{pmatrix}.$$

As long as the threshold $\mu$ is small, the neuron's output is halfway between its maximal and minimal values whenever its input is zero, i.e. when

$$(x_c, y_c, f)\sum_{i=1}^{m}\left[\tilde{\mathbf{B}}^t(x_i, y_i)\begin{pmatrix} C_{l,2i-1}^{\perp}(\mathbf{T}_j) \\ C_{l,2i}^{\perp}(\mathbf{T}_j) \end{pmatrix}\right] = 0.$$

This is the equation of a line in the $(x_c, y_c)$ plane. The neuron's selectivity for rotations reverses along this line. A similar equation holds expansion/contraction selectivity.

Now, what would the neuron's selectivity look like, if we had not restricted the eye movements to the case of the fixation of an object. The responses of a neuron that is constructed following the unconstrained version of the algorithm, as described in section 2, is shown in Fig. 2. There is no selectivity for clockwise versus counterclockwise rotations at all, since both patterns elicit the same response everywhere in the visual field. Inside the receptive field the neuron favors contractions and shifts towards the upper left ($\Phi = 150$ deg.). It can thus be regarded as a double component cell. To understand the absence of rotational selectivity we have to calculate the whole rotational optic flow pattern $\Theta_{\text{rot}}$ by inserting $\mathbf{T}$

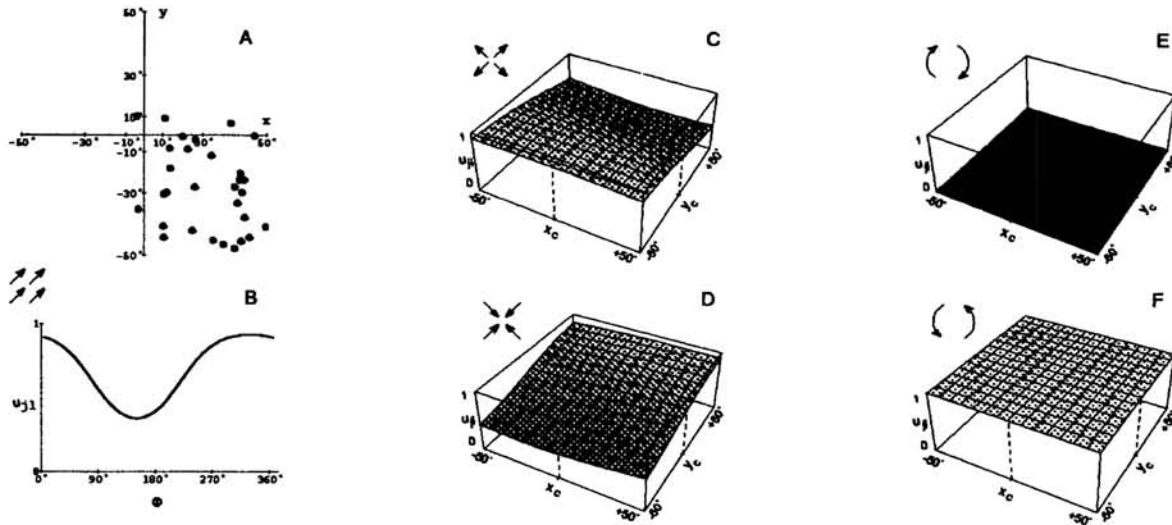

Figure 3: Predominantly Rotation Selective Neuron ("Single Component Cell")

and $\Omega$ from eq. (7) into eq. (3). $\mathbf{C(T)}$ becomes

$$
\mathbf{C}(0) = \begin{pmatrix} 0 & \cdots & 0 & \mathbf{B}(x_1, y_1) \\ \vdots & \ddots & \vdots & \vdots \\ 0 & \cdots & 0 & \mathbf{B}(x_m, y_m) \end{pmatrix} .
$$

Denoting the three rightmost column vectors of $\mathbf{C(T)}$ by $\mathbf{B}_1$, $\mathbf{B}_2$, and $\mathbf{B}_3$ we find

$$
\Theta_{\mathrm{rot}} = \frac{\pm\Omega}{\sqrt{x_c^2 + y_c^2 + f^2}}(x_c\mathbf{B}_1 + y_c\mathbf{B}_2 + f\mathbf{B}_3).
$$

Comparison to $\mathbf{C(T)}$, eq. (4), shows that $\Theta_{\mathrm{rot}}$ can be written as a linear combination of column vectors of $\mathbf{C(T)}$. Thus $\Theta_{\mathrm{rot}}$ lies in the range of $\mathbf{C(T)}$ and is orthogonal to $\mathbf{C}^\perp(\mathbf{T})$, so that $\Theta_{\mathrm{rot}}\,\mathbf{C}_l^\perp(\mathbf{T}_j) = 0$ for all $j$ and $l$. From eqs. (5) and (6) it follows, that the neuron's response to any rotational pattern is always $u_{jl} = g(-\mu)$.

The last type of eye movements we want to consider is that of a general frontoparallel rotation, which is defined by $\Omega_Z = 0$. In addition to the fixation of a stationary object, frontoparallel rotations also include smooth pursuit eye movements necessary for the fixation of a moving object. Inserting $\Omega_Z = 0$ into eq. (2) gives

$$
\hat{\theta}(x, y) = \frac{1}{Z(x, y)}\mathbf{A}(x, y)\mathbf{T} + \hat{\mathbf{B}}(x, y)\begin{pmatrix} \Omega_X \\ \Omega_Y \end{pmatrix}
$$

with

$$
\hat{\mathbf{B}}(x, y) = \begin{pmatrix} xy/f & -(f + x^2/f) \\ f + y^2/f & -xy/f \end{pmatrix}
$$

now being a $2 \times 2$ matrix, so that $\mathbf{C(T)}$, eq. (4), becomes a $2m \times (m + 2)$ matrix $\hat{\mathbf{C}}(\mathbf{T})$. A neuron that is constructed using $\hat{\mathbf{C}}(\mathbf{T})$ can be seen in Fig. 3. It best responds to counterclockwise rotational patterns showing complete position invariance over the visual field. The neuron is much less selective to expansions and unidirectional shifts, since

the responses never reach saturation. It therefore resembles a single component rotation selective cell. The position invariant behavior can again be explained by looking at the rotational optic flow pattern. Using the same argument as above, one can show that the neuron's input is zero whenever $\Omega_Z$ vanishes, i.e. when the rotational axis lies in the $(X, Y)$–plane. Then the flow pattern becomes

$$\Theta_{\mathrm{rot}} = \frac{\pm\Omega}{\sqrt{x_c^2 + y_c^2 + f^2}}(x_c\hat{\mathbf{B}}_1 + y_c\hat{\mathbf{B}}_2),$$

and is an element of the range of $\hat{\mathbf{C}}(\mathbf{T}_j)$. The $(X, Y)$–plane thus splits the space of all rotational axes into two half spaces, one in which the neuron's input is always positive and one in which it is always negative. Clockwise rotations are characterized by $\Omega_Z > 0$ and hence all lie in the same half space, while counterclockwise rotations lie in the other. As a result the neuron is exclusively excited by one mode of rotation in all of the visual field.

## 4 Conclusion

Our neural network model for the detection of ego–motion proposes a computational map of heading directions. A similar map could be contained in area MSTd of monkey visual cortex. Cells in MSTd exhibit a varying degree of selectivity for basic optic flow patterns, but often show a substantial indifference towards the spatial position of a stimulus. By using a population encoding of the heading directions, individual neurons in the model exhibit similar position invariant responses within large parts of the visual field. Different neuronal selectivities found in MSTd can be modelled by assuming specializations pertaining to different types of eye movements. Consistent with experimental findings the position invariance of the model neurons is largest in the single component cells and less developed in the double and triple component cells.

## Footnotes

*Present address: Neurobiologie, ND7, Ruhr–Universität Bochum, 4630 Bochum, Germany.

## References

Allman, J. M. and Kaas, J. H. 1971. *Brain Res.* **31**, 85–105.

Andersen, R., Graziano, M., and Snowden, R. 1990. *Soc. Neurosci. Abstr.* **16**, 7.

Brenner, E. and Rauschecker, J. P. 1990. *J. Physiol.* **423**, 641–660.

Duffy, C. J. and Wurtz, R. H. 1991. *J. Neurophysiol.* **65**(6), 1329–1359.

Gibson, J. J. 1950. *The Perception of the Visual World*. Houghton Mifflin, Boston.

Heeger, D. J. and Jepson, A. 1992. *Int. J. Comp. Vis.* **7**(2), 95–117.

Lappe, M. and Rauschecker, J. P. 1993. *Neural Computation (in press)*.

Maunsell, J. H. R. and Van Essen, D. C. 1983. *J. Neurophysiol.* **49**(5), 1127–1147.

Orban, G. A., Lagae, L., Verri, A., Raiguel, S., Xiao, D., Maes, H., and Torre, V. 1992. *Proc. Nat. Acad. Sci.* **89**, 2595–2599.

Rauschecker, J. P., von Grünau, M. W., and Poulin, C. 1987. *J. Neurosci.* **7**(4), 943–958.

Rieger, J. H. and Toet, L. 1985. *Biol. Cyb.* **52**, 377–381.

Stone, L. S. and Perrone, J. A. 1991. In *Soc. Neurosci. Abstr.* **17**, 857.

Tanaka, K. and Saito, H.-A. 1989. *J. Neurophysiol.* **62**(3), 626–641.

Wang, H. T., Mathur, B. P. and Koch, C. 1989. *Neural Computation* **1**, 92–103.

Warren, W. H. Jr., and Hannon, D. J. 1988. *Nature* **336**, 162–163.
